# Robust Regression with Twinned Gaussian Processes

**Andrew Naish-Guzman & Sean Holden**
Computer Laboratory
University of Cambridge
Cambridge, CB3 0FD. United Kingdom
{agpn2,sbh11}@cl.cam.ac.uk

## Abstract

We propose a Gaussian process (GP) framework for robust inference in which a GP prior on the mixing weights of a two-component noise model augments the standard process over latent function values. This approach is a generalization of the mixture likelihood used in traditional robust GP regression, and a specialization of the GP mixture models suggested by Tresp [1] and Rasmussen and Ghahramani [2]. The value of this restriction is in its tractable expectation propagation updates, which allow for faster inference and model selection, and better convergence than the standard mixture. An additional benefit over the latter method lies in our ability to incorporate knowledge of the noise domain to influence predictions, and to recover with the predictive distribution information about the outlier distribution via the gating process. The model has asymptotic complexity equal to that of conventional robust methods, but yields more confident predictions on benchmark problems than classical heavy-tailed models and exhibits improved stability for data with clustered corruptions, for which they fail altogether. We show further how our approach can be used without adjustment for more smoothly heteroscedastic data, and suggest how it could be extended to more general noise models. We also address similarities with the work of Goldberg et al. [3].

## 1 Introduction

Regression data are often modelled as noisy observations of an underlying process. The simplest assumption is that all noise is independent and identically distributed (i.i.d.) zero-mean Gaussian, such that a typical set of samples appears as a cloud around the latent function. The Bayesian framework of Gaussian processes [4] is well-suited to these conditions, for which all computations remain tractable (see figure 1a). Furthermore, the Gaussian noise model enjoys the theoretical justification of the central limit theorem, which states that the sum of sufficiently many i.i.d. random variables of finite variance will be distributed normally. However, only rarely can perturbations affecting data in the real world be argued to have originated in the addition of many i.i.d. sources. The random component in the signal may be caused by human or measurement error, or it may be the manifestation of systematic variation invisible to a simplified model. In any case, if ever there is the possibility of encountering small quantities of highly implausible data, we require *robustness*, i.e. a model whose predictions are not greatly affected by outliers.

Such demands render the standard GP inappropriate: the light tails of the Gaussian distribution cannot explain large non-Gaussian deviations, which either skew the mean interpolant away from the majority of the data, or force us to infer an unreasonably large (global) noise variance (see figure 1b). Robust methods use a heavy-tailed likelihood to allow the interpolant effectively to favour smoothness and ignore such erroneous data. Figure 1c shows how this can be achieved using a two-component noise model

$$p(y_n|f_n) = (1 - \epsilon)\mathcal{N}\left(y_n\,;\,f_n\,,\,\sigma_R^2\right) + \epsilon\mathcal{N}\left(y_n\,;\,f_n\,,\,\sigma_O^2\right), \tag{1}$$

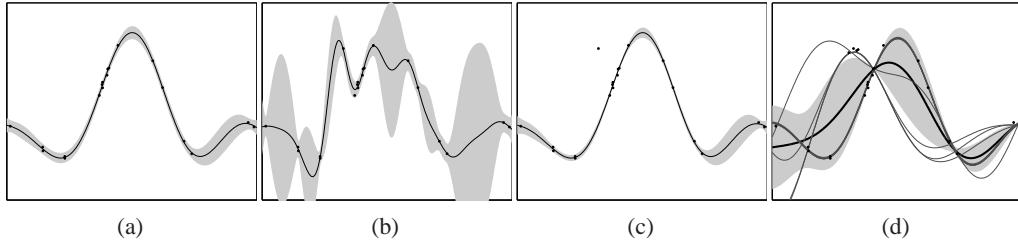

Figure 1: Black dots show noisy samples from the `sinc` function. In panels (a) and (b), the behaviour of a GP with a Gaussian noise assumption is illustrated; the shaded region shows 95% confidence intervals. The presence of a single outlier is highly influential in this model, but the heavy-tailed likelihood (1) in panel (c) is more resilient. Unfortunately, even this model fails for the cluster of outliers in panel (d). Here, grey lines show ten repeated runs of the EP inference algorithm, while the black line and shaded region are their averaged mean and confidence intervals respectively—grossly at odds with those of the latent generative model.

in which observations $y_n$ are Gaussian corruptions of $f_n$, being drawn with probability $\epsilon$ from a large variance outlier distribution ($\sigma_O^2 \gg \sigma_R^2$). Inference in this model is tractable, but impractical for all but the smallest problems due to the exponential explosion of terms in products of (1).

In this paper, we address the more fundamental GP assumption of i.i.d. noise. Our research is motivated by observing how the predictive distribution suffers for heavy-tailed models when outliers appear in bursts: figure 1d replicates figure 1c, but introduces an additional three outliers. All parameters were taken from the optimal solution to (c), but even without the challenge of hyperparameter optimization there is now considerable uncertainty in the posterior since the competing interpretations of the cluster as signal or noise have similar posterior mass. Viewed another way, the tails of the *effective* log likelihood of four clustered observations have approximately one-quarter the weight of a single outlier, so the magnitude of the posterior peak associated with the robust solution is comparably reduced. One simple remedy is to make the tails of the likelihood heavier. However, since the noise model is global, this has ramifications across the entire data space, potentially causing underfitting elsewhere when real data are relegated to the tails. We can establish an optimal choice for the parameters by gradient ascent on the marginal likelihood, but it is entirely possible that no single setting will be universally satisfactory.

The model introduced in this paper, which we call the *twinned Gaussian process* (TGP), generalizes the noise model (1) by using a GP gating function to choose between the "real" and "outlier distributions": in regions of confidence, the tails can be made very light, encouraging the interpolant to hug the data points tightly; more dubious observations can be treated appropriately by broadening the noise distribution in their vicinity. Our model is also a specialization of the GP mixtures proposed by Tresp [1] and Rasmussen and Ghahramani [2]; indeed, the latter automatically infers the correct number of components to use. One may therefore wonder what can possibly be gained by restricting ourselves to a comparatively simple architecture. The answer is in the computational overhead required for the different approaches, since these more general models require inference by Monte Carlo methods. We argue that the two-component mixture is often a sensible distribution for modelling real data, with a natural interpretation and the heavy tails required for robustness; its weaknesses are exposed primarily when the noise distribution is not homoscedastic. The TGP largely solves this problem, and allows inference by an efficient expectation propagation (EP) [5] procedure (rather than resorting to more heavy duty Monte Carlo methods). Hence, provided a two-component mixture is likely to reflect adequately the noise on our data, the TGP will give similar results to the generalized mixtures mentioned above, but at a fraction of the cost.

Goldberg et al. [3] suggest an approach to input-dependent noise in the spirit of the TGP, in which the log variance on observations is itself modelled as a GP (the logarithm since noise variance is a non-negative property). Inference is again analytically intractable, so Gibbs sampling is used to generate noise vectors from the posterior distribution by alternately fitting the signal process and fitting the noise process. A further stage of Gibbs sampling is required at each test point to estimate the predictive variance, making testing rather slow. Model selection is even slower, and the Metropolis-Hastings algorithm is suggested for updating hyperparameters.

## 2 Twinned Gaussian processes

Given a domain $\mathcal{X}$ and covariance function $K(\cdot, \cdot) \in \mathcal{X} \times \mathcal{X} \rightarrow \mathbb{R}$, a Gaussian process (GP) over the space of real-valued functions of $\mathcal{X}$ specifies the joint distribution at any finite set $\mathbf{X} \subset \mathcal{X}$:

$$p(\mathbf{f}|\mathbf{X}) = \mathcal{N}(\mathbf{f} \, ; \, \mathbf{0} \, , \, \mathbf{K}_f) \, ,$$

where the $\mathbf{f} = \{f_n\}_{n=1}^N$ are (latent) values associated with each $\mathbf{x}_n \in \mathbf{X}$, and $\mathbf{K}_f$ is the *Gram matrix*, the evaluation of the covariance function at all pairs $(\mathbf{x}_i, \mathbf{x}_j)$. We apply Bayes' rule to obtain the posterior distribution over the $\mathbf{f}$, given the observed $\mathbf{X}$ and $\mathbf{y}$, which with the assumption of i.i.d. Gaussian corrupted observations is also normally distributed. Predictions at $\mathbf{X}_\star$ are made by marginalizing over $\mathbf{f}$ in the (Gaussian) joint $p(\mathbf{f}, \mathbf{f}_\star|\mathbf{X}, \mathbf{y}, \mathbf{X}_\star)$. See [6] for a thorough introduction.

Robust GP regression is achieved by using a *leptokurtic* likelihood distribution, i.e. one whose tails have more mass than the Gaussian. Common choices are the Laplace (or double exponential) distribution, Student's t distribution, and the mixture model (1). In product with the prior, a heavy-tailed likelihood over an outlying observation does not exert the strong pull on the posterior witnessed with a light-tailed noise model. Kuss [7] describes how inference can be performed for all these likelihoods, and establishes that in many cases their performance is broadly comparable. Since it bears closest resemblance to the twinned GP, we are particularly interested in the mixture; however, in section 4, we include results for the Laplace model: it is the heaviest-tailed log concave distribution, which guarantees a unimodal posterior and allows more reliable EP convergence. In any case, all such methods make a *global* assumption about the noise distribution, and it is where this is inappropriate that our model is most beneficial.

The graphical model for the TGP is shown in figure 2b. We augment the standard process over $\mathbf{f}$ with another GP over a set of variables $\mathbf{u}$; this acts as a gating function, probabilistically dividing the domain between the real and outlier components of the noise model

$$p(y_n|f_n) = \sigma(u_n)\mathcal{N}\left(y_n \, ; \, f_n \, , \, \sigma_R^2\right) + \sigma(-u_n)\mathcal{N}\left(y_n \, ; \, f_n \, , \, \sigma_O^2\right) , \qquad (2)$$

$$\text{where} \quad \sigma(u_n) \doteq \int_{-\infty}^{u_n} \mathcal{N}(z \, ; \, 0 \, , \, 1) \, \mathrm{d}z.$$

In the TGP likelihood, we therefore mix two forms of Gaussian corruption, one strongly peaked at the observation, the other a broader distribution which provides the heavy tails, in proportion determined by $u(\mathbf{x})$. This makes intuitive sense; crucially to us, it retains the advantage of tractability with respect to EP updates. The two priors may have quite different covariance structure, reflecting our different beliefs about correlations in the signal and in the noise domain. In addition, we accommodate prior beliefs about the prevalence of outliers with a non-zero mean process on $\mathbf{u}$,

$$p(\mathbf{u}|X) = \mathcal{N}(\mathbf{u} \, ; \, \mathbf{m}_u \, , \, \mathbf{K}_u) \qquad\qquad p(\mathbf{f}|X) = \mathcal{N}(\mathbf{f} \, ; \, \mathbf{0} \, , \, \mathbf{K}_f) \, .$$

Our model can be understood as lying between two extremes: observe that we recover the heavy-tailed (mixture of Gaussians) GP by forcing absolute correlation in $\mathbf{u}$ and adjusting the mean of the $\mathbf{u}$-process to $m_u = \sigma^{-1}(1-e)$; conversely, if we remove all correlations in $\mathbf{u}$, we return to a standard mixture model where independently we must decide to which component an input belongs.

## 3 Inference

We begin with a very brief account of EP; for more details, see [5, 8]. Suppose we have an intractable distribution over $\mathbf{f}$ whose unnormalized form factorizes into a product of terms, such as a dense Gaussian prior $t_0(\mathbf{f}, \mathbf{u})$ and a series of independent likelihoods $\{t_n(y_n|f_n, u_n)\}_{n=1}^N$. EP constructs the approximate posterior as a product of scaled *site functions* $\tilde{t}_n$. For computational tractability, these sites are usually chosen from an exponential family with natural parameters $\boldsymbol{\theta}$, since in this case their product retains the same functional form as its components. The Gaussian $(\boldsymbol{\mu}, \boldsymbol{\Sigma})$ has a natural parameterization $(\mathbf{b}, \boldsymbol{\Pi}) = (\boldsymbol{\Sigma}^{-1}\boldsymbol{\mu}, -\frac{1}{2}\boldsymbol{\Sigma}^{-1})$. If the prior is of this form, its site function is exact:

$$p(\mathbf{f}, \mathbf{u}|\mathbf{y}) = \frac{1}{Z} t_0(\mathbf{f}, \mathbf{u}) \prod_{n=1}^N t_n(y_n|f_n, u_n) \approx q(\mathbf{f}; \boldsymbol{\theta}) = t_0(\mathbf{f}, \mathbf{u}) \prod_{n=1}^N z_n \tilde{t}_n(f_n, u_n; \theta_n), \qquad (3)$$

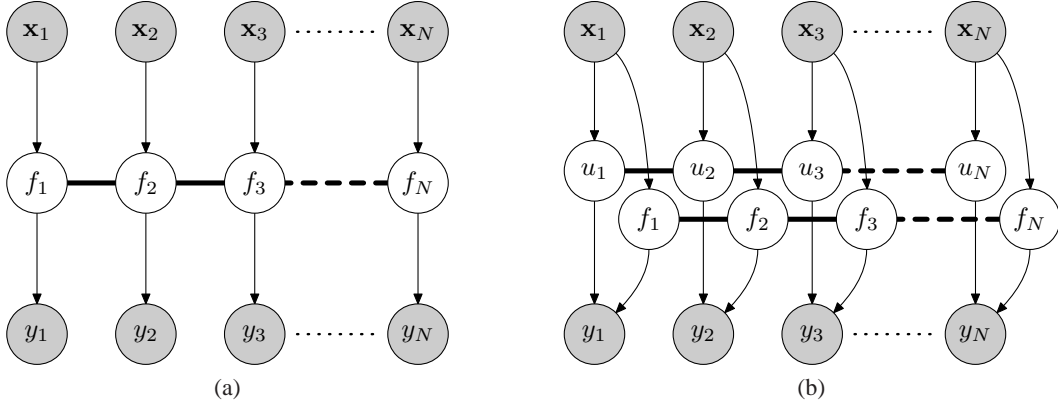

Figure 2: In panel (a) we show a graphical model for the Gaussian process. The data ordinates are $\mathbf{x}$, observations $\mathbf{y}$, and the GP is over the latent $\mathbf{f}$. The bold black lines indicate a fully-connected set. Panel (b) shows a graphical model for the *twinned Gaussian process* (TGP), in which an auxiliary set of hidden variables $\mathbf{u}$ describes the noisiness of the data.

where $Z$ is the marginal likelihood and $z_n$ are the scale parameters. Ideally, we would choose $\boldsymbol{\theta}$ at the global minimum of some divergence measure $d(p\|q)$, but the necessary optimization is usually intractable. EP is an iterative procedure that finds a minimizer of $\mathsf{KL}\big(p(\mathbf{f},\mathbf{u}|\mathbf{y})\|q(\mathbf{f},\mathbf{u};\boldsymbol{\theta})\big)$ on a pointwise basis: at each iteration, we select a new site $n$, and from the product of the *cavity* distribution formed by the current marginal with the omission of that site, and the true likelihood term $t_n$, we obtain the so-called *tilted* distribution $q^n(f_n,u_n;\boldsymbol{\theta}^{\backslash n})$. A simpler optimization $\min_{\theta_n} \mathsf{KL}\big(q^n(f_n,u_n;\boldsymbol{\theta}^{\backslash n})\|q(f_n,u_n;\boldsymbol{\theta})\big)$ then fits only the parameters $\theta_n$: this is equivalent to *moment matching* between the two distributions, with scale $z_n$ chosen to match the zeroth-order moments. After each site update, the moments at the remaining sites are liable to change, and several iterations may be required before convergence.

The priors over $\mathbf{u}$ and $\mathbf{f}$ are independent, but we expect correlations in the posterior after conditioning on observations. To understand this, consider a single observation $(\mathbf{x}_n,y_n)$; in principle, it admits two explanations corresponding to its classification as either "outlier" or as "real" data: in general terms, either $u_n > 0$ and $f_n \approx y_n$, or $u_n < 0$ and $f_n$ respects the global structure of the signal. A diagram to assist the visualization of the behaviour of the posterior is provided in figure 3.

Now, recall that the prior over $\mathbf{u}$ and $\mathbf{f}$ is

$$p\left(\left[\begin{array}{c}\mathbf{u}\\\mathbf{f}\end{array}\right]\bigg|X\right) = \mathcal{N}\left(\left[\begin{array}{c}\mathbf{u}\\\mathbf{f}\end{array}\right];\left[\begin{array}{c}\mathbf{m}_u\\\mathbf{0}\end{array}\right],\left[\begin{array}{cc}\mathbf{K}_u & \mathbf{0}\\\mathbf{0} & \mathbf{K}_f\end{array}\right]\right)$$

and the likelihood factorizes into a product of terms (2); our site approximations $\tilde{t}_n$ are therefore Gaussian in $(f_n,u_n)$. Of importance for EP are the moments of the tilted distribution which we seek to match. These are most easily obtained by differentiation of the zeroth moments $Z_R$ and $Z_O$ of each component. We find

$$Z_R = \iint_{f,u} \sigma(u)\mathcal{N}(y;f,\sigma_R^2)\mathcal{N}\left(\left[\begin{array}{c}u\\f\end{array}\right];\boldsymbol{\mu},\boldsymbol{\Sigma}\right)\mathrm{d}u\mathrm{d}f = \int_0^\infty \mathcal{N}\left(\left[\begin{array}{c}z\\y\end{array}\right];\boldsymbol{\mu},\left[\begin{array}{cc}1 & 0\\0 & \sigma_R^2\end{array}\right]+\boldsymbol{\Sigma}\right)\mathrm{d}z;$$

writing the inner Gaussian as $\mathcal{N}\left(\left[\begin{array}{c}z_n\\y_n\end{array}\right];\left[\begin{array}{c}\mu_u\\\mu_f\end{array}\right],\left[\begin{array}{cc}A & C\\C & B_R\end{array}\right]\right)$, $Z^R = \mathcal{N}(y;\mu_f,B_R)\,\sigma(q)$,

$$\text{where} \quad q = \frac{\mu_u + \frac{C}{B_R}(y-\mu_f)}{\sqrt{A - \frac{C^2}{B_R}}}.$$

The integral for the outlier component is similar; $Z_O = \mathcal{N}(y;\mu_f,B_O)\,\sigma(-q)$. With partial derivatives $\frac{\partial\log Z}{\partial\boldsymbol{\mu}}$ and $\frac{\partial^2\log Z}{\partial\boldsymbol{\mu}\boldsymbol{\mu}^T}$ we are equipped for EP; algorithmic details appear in Seeger's note [8]. For efficiency, we make rank-two updates of the full approximate covariance on $(\mathbf{f},\mathbf{u})$ during the EP loop, and refresh the posterior at the end of each cycle to avoid loss of precision.

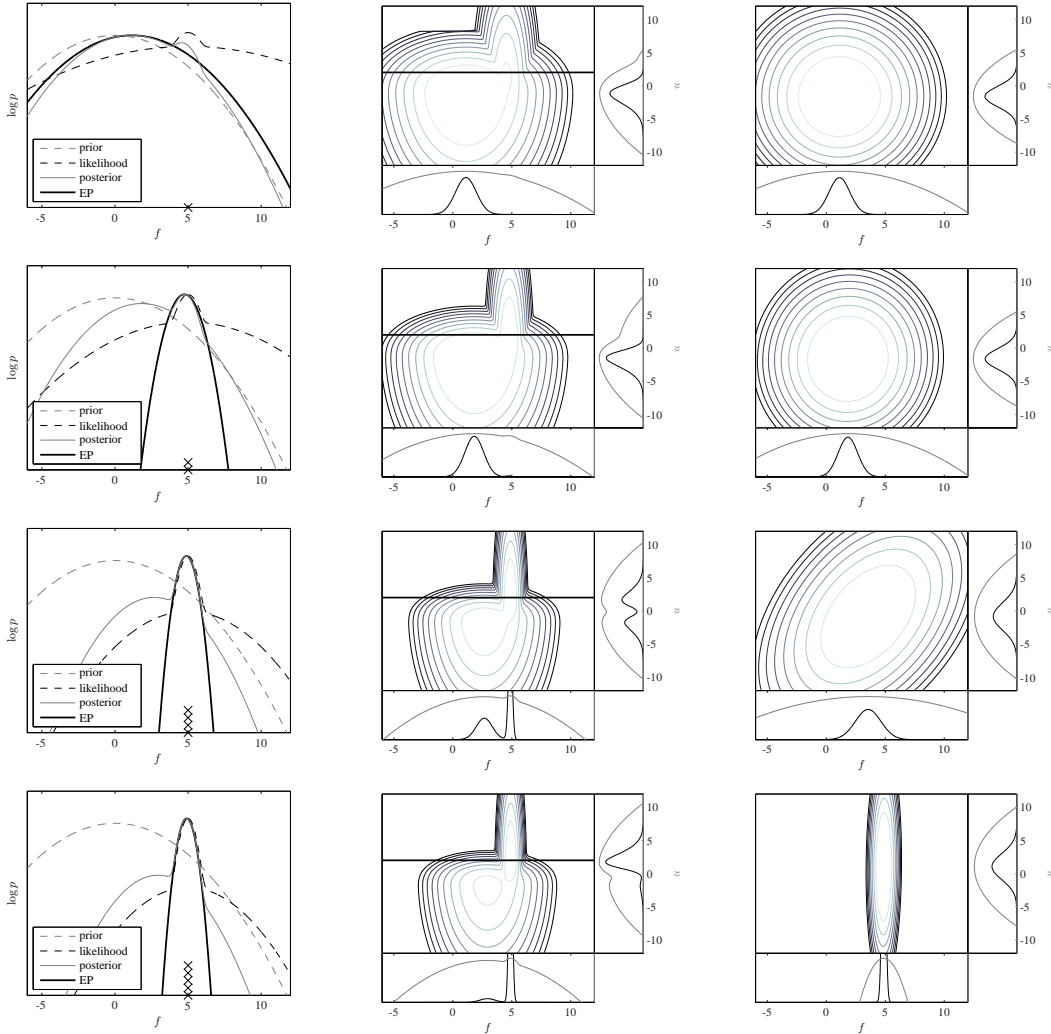

Figure 3: Using the twinned Gaussian process provides a natural resilience against clustered noisy data. The left-hand column illustrates the behaviour of a fixed heavy-tailed likelihood for one, two, four and five repeated observations at $f = 5$. (Outliers in real data are not necessarily so tightly packed, but the symmetry of this approximation allows us to treat them as a single unit: by "posterior", for example, we mean the a posteriori belief in *all* the observations' (identical) latent $f$.) The context is provided by the prior, which gives 95% confidence to data around $f = 0 \pm 2$. The top-left box illustrates how the influence of isolated outliers is mitigated by the standard mixture. However, a repeated observation (box two on the left) causes the EP solution to collapse onto the spike at the data (the log scale is deceptive: the second peak contributes only about 8% of the posterior mass). The twinned GP better preserves the marginal distribution of $f$ by maintaining a joint distribution over both $f$ and $u$: in the second and third columns respectively are contours of the true log joint (we use a broad zero-mean prior on $u$) and that inferred by EP, together with the marginal posterior over $f$. Only with a fifth observation—final box—is the context of $f$ essentially overruled by the TGP approximation. The thick bar in the central column marks the cross-section corresponding to the unnormalized posterior from column one.

## 3.1 Predictions

If the outlier component describes nuisance noise that should be eliminated, we require at test inputs $\mathbf{x}_\star$ only the marginal distribution $p(f_\star|\mathbf{x}_\star, X, \mathbf{y})$, obtained by marginalizing over $\mathbf{u}$ in the full (approximate) posterior

$$\mathcal{N}\left(\begin{bmatrix} \mathbf{u} \\ \mathbf{f} \end{bmatrix} ; \begin{bmatrix} \hat{\boldsymbol{\mu}}_u \\ \hat{\boldsymbol{\mu}}_f \end{bmatrix}, \begin{bmatrix} \hat{\boldsymbol{\Sigma}}_{uu} & \hat{\boldsymbol{\Sigma}}_{uf} \\ \hat{\boldsymbol{\Sigma}}_{fu} & \hat{\boldsymbol{\Sigma}}_{ff} \end{bmatrix}\right):$$

$$p(f_\star|\mathbf{x}_\star, X, \mathbf{y}) = \int p(f_\star|\mathbf{x}_\star, \mathbf{f}) p(\mathbf{f}|X, \mathbf{y}) \mathrm{d}\mathbf{f}$$

$$\approx \mathcal{N}\left(f_\star ; \mathbf{k}_{f\star}^T \mathbf{K}_f^{-1} \hat{\boldsymbol{\mu}}_f , k_{\star\star}^f - \mathbf{k}_{f\star}^T \mathbf{K}_f^{-1} \mathbf{k}_{f\star} + \mathbf{k}_{f\star}^T \mathbf{K}_f^{-1} \hat{\boldsymbol{\Sigma}}_{ff} \mathbf{K}_f^{-1} \mathbf{k}_{f\star}\right).$$

The noise process may itself be of interest, in which case we need to marginalize over both $u_\star$ and $f_\star$ in

$$p(y_\star|\mathbf{x}_\star, X, \mathbf{y}) = \iint p\left(y_\star \middle| \mathbf{x}_\star, \begin{bmatrix} \mathbf{u} \\ \mathbf{f} \end{bmatrix}\right) p\left(\begin{bmatrix} \mathbf{u} \\ \mathbf{f} \end{bmatrix} \middle| X, \mathbf{y}\right) \mathrm{d}\mathbf{u}\mathrm{d}\mathbf{f}$$

$$\approx \iiiint p\left(y_\star \middle| \mathbf{x}_\star, \begin{bmatrix} u_\star \\ f_\star \end{bmatrix}\right) p\left(\begin{bmatrix} u_\star \\ f_\star \end{bmatrix} \middle| \begin{bmatrix} \mathbf{u} \\ \mathbf{f} \end{bmatrix}\right) \mathcal{N}\left(\begin{bmatrix} \mathbf{u} \\ \mathbf{f} \end{bmatrix} ; \hat{\boldsymbol{\mu}} , \hat{\boldsymbol{\Sigma}}\right) \mathrm{d}u_\star \mathrm{d}f_\star \mathrm{d}\mathbf{u}\mathrm{d}\mathbf{f}.$$

This distribution is no longer Gaussian, but its moments may be recovered easily by the same method used to obtain moments of the tilted distribution.

EP provides in addition to the approximate moments of the posterior distribution an estimate of the marginal likelihood and its derivatives with respect to kernel hyperparameters. Again, we refer the interested reader to the algorithm presented in [8], adding here only that our implementation uses log noise values on $(\sigma_R^2, \sigma_O^2)$ to allow for their unconstrained optimization.

## 3.2 Complexity

The EP loop is dominated by the rank-two updates of the covariance. Each such update is $\mathcal{O}\left((2N)^2\right)$, making every $N$ iterations $\mathcal{O}(4N^3)$. The posterior refresh is $\mathcal{O}(8N^3)$ since it requires the inverse of a $2N \times 2N$ positive semi-definite matrix, most efficiently achieved through Cholesky factorization (this Cholesky factor can be retained for use in calculating the approximate log marginal likelihood). The total number of loops required for convergence of EP is typically independent of $N$, and can be upper bounded by a small constant, say 10, making the entire inference process $\mathcal{O}(8N^3) = \mathcal{O}(N^3)$. Thus, our algorithm has the same limiting time complexity as i.i.d. robust regression by EP, which admittedly masks the larger coefficient that appears in approximating both $\mathbf{u}$ and $\mathbf{f}$ simultaneously. Additionally, the body of the EP loop is slightly slower, since the precision matrix in a standard GP can be obtained with a single division, whereas our model requires the inversion of a $2 \times 2$ matrix.

# 4 Experiments

We identify two general noise characteristics for which our model may be suitable. The first is when the outlying observations can appear in clusters: we saw in figure 1d how these occurrences affect the standard mixture model. In fact the problem is quite severe, since the multimodality of the posterior impedes the convergence of EP, while the possibility of conflicting gradient information at the optima hampers procedures for evidence maximization. In figure 4 we illustrate how the TGP succeeds where the mixture and Laplace models fail; note how the mean process on $\mathbf{u}$ falls sharply in the contaminated regions. This is a stable solution, and hyperparameters can be fit reliably.

A data set which exhibits the superior predictive modelling of the TGP in a domain where robust methods can also expect to perform well is provided by Kuss [7] in a variation on a set of Friedman [9]. The samples are drawn from a function of ten-dimensional vectors $\mathbf{x}$ which depend only on the first five components:

$$f(\mathbf{x}) = 10\sin(\pi x_1 x_2) + 20(x_3 - 0.5)^2 + 10x_4 + 5x_5.$$

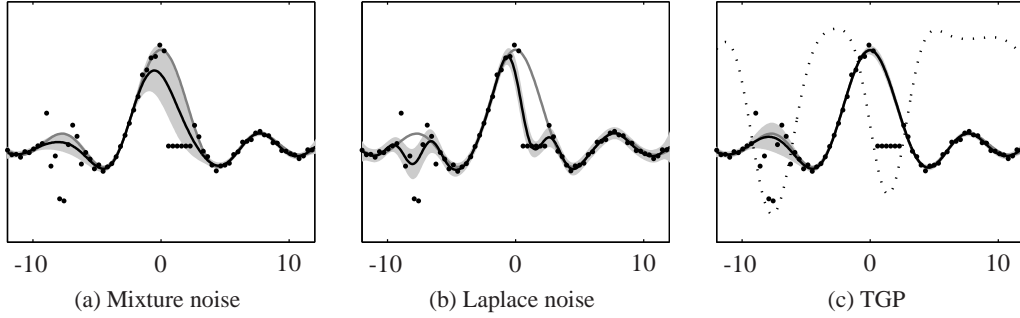

(a) Mixture noise          (b) Laplace noise          (c) TGP

Figure 4: The corruptions are i.i.d. around $x = -10$, and highly correlated near $x = 0$.

We generated ten sets of 90 training examples and 10000 test examples by sampling $\mathbf{x}$ uniformly in $[0, 1]^{10}$, and adding to the training data noise $\mathcal{N}(0, 1)$. In our first experiment, we replicated the procedure of [7]: ten training points were added at random with outputs sampled from $\mathcal{N}(15, 9)$ (a value likely to lie in the same range as $f$). The results appear as Friedman (1) in figure 5. Observe that the r.m.s. error for the robust methods is similar, but the TGP is able to fit the variance far more accurately. In a second experiment, the training set was augmented with two Gaussian clusters each of five noisy observations. The cluster centres were drawn uniformly in $[0, 1]^{10}$, with variance fixed at $10^{-3}$. Output values were then drawn from $\mathcal{N}(0, 1)$ for all ten points, to give highly correlated values distant from the underlying function (Friedman (2)). Now the TGP excels where the other methods offer no improvement on the standard GP; it also yields very confident predictions (cf. Friedman (1)), because once the outliers have been accounted for there are fewer corrupted regions; furthermore, estimates of where the data are corrupted can be recovered by considering the process on $\mathbf{u}$. In both experiments, the training data were renormalized to zero mean and unit variance, and throughout, we used the anisotropic squared exponential for the $\mathbf{f}$ process (implementing so-called relevance determination), and an isotropic version for $\mathbf{u}$. The approximate marginal likelihood was maximized on three to five randomly initialized models; we chose for testing the most favoured.

The second domain of application is when the noise on the data is believed a priori to be a function of the input (i.e. heteroscedastic). The twinned GP can simulate this changing variance by modulating the $\mathbf{u}$ process, allocating varying weight to the two components. By way of example, the behaviour for the one-dimensional motorcycle set [10] is shown in fig. 5c. However, since the input-dependent noise is not modelled directly, there are two notable dangers associated with this approach: first, the predictive variance saturates when all weight has been apportioned to one or other component; second, the "outlier" component can dominate the variance estimates of the mixture. This is particularly problematic when variance on the data ranges over several orders of magnitude, such that the "outlier" width must be comparably broader than that of the "real" component. In such cases, only with extreme values of $\mathbf{u}$ can the smallest errors be predicted, but in consequence the process tends to sweep precipitately through the region of sensitivity where variance predictions can be made accurately. To circumvent these problems we might employ the warped GP [11] to rescale the process on $\mathbf{u}$ in a supervised manner, but we do not explore these ideas further here.

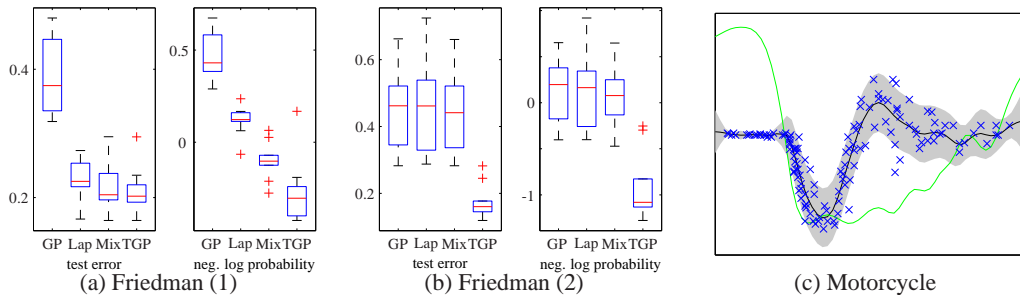

(a) Friedman (1)          (b) Friedman (2)          (c) Motorcycle

Figure 5: Results for the Friedman data, and the predictions of the TGP on the motorcycle set.

## 5 Extensions

With prior knowledge of the nature of corruptions affecting the signal, we can seek to model the noise distribution more accurately, for example by introducing a compound likelihood for the outlier component $p_O(y_n|f_n) = \sum_j \alpha_j \mathcal{N}\left(y_n\,;\,\mu_j(f_n)\,,\,\sigma_j^2\right), \sum_j \alpha_j = 1$. This constrains the relative weight of outlier corruptions to be constant across the entire domain. A richer alternative is provided by extending the single $\mathbf{u}$-process on noise to a series $\mathbf{u}^{(1)}, \mathbf{u}^{(2)}, \ldots, \mathbf{u}^{(\nu)}$ of noise processes, and broadening the likelihood function appropriately. For example, with $\nu = 2$, we may write

$$
\begin{aligned}
p(y_n|f_n, u_n^{(1)}, u_n^{(2)}) = \sigma(u_n^{(1)})\mathcal{N}\left(y_n\,;\,f_n\,,\,\sigma_R^2\right) + \\
\sigma(-u_n^{(1)})\sigma(u_n^{(2)})\mathcal{N}\left(y_n\,;\,f_n\,,\,\sigma_{O_1}^2\right) + \\
\sigma(-u_n^{(1)})\sigma(-u_n^{(2)})\mathcal{N}\left(y_n\,;\,f_0\,,\,\sigma_{O_2}^2\right). \quad (4)
\end{aligned}
$$

In the former case, the preceding analysis applies with small changes: each component of the outlier distribution contributes moments independently. The second model introduces significant computational difficulty: firstly, we must maintain a posterior distribution over $\mathbf{f}$ and all $\nu$ $\mathbf{u}$s, yielding space requirements $\mathcal{O}(N(\nu + 1))$ and time complexity $\mathcal{O}(N^3(\nu + 1)^3)$. More importantly, the requisite moments needed in the EP loop are now intractable, although an inner EP loop can be used to approximate them, since the product of $\sigma$s behaves in essence like the standard model for GP classification. We omit details, and defer experiments with such a model to future work.

## 6 Conclusions

We have presented a method for robust GP regression that improves upon classical approaches by allowing the noise variance to vary in the input space. We found improved convergence on problems which upset the standard mixture model, and have shown how predictive certainty can be improved by adopting the TGP even for problems which do not. The model also allows an arbitrary process on $\mathbf{u}$, such that specialized prior knowledge could be used to drive the inference over $\mathbf{f}$ to respecting regions which may otherwise be considered erroneous. A generalization of our ideas appears as the mixture of GPs [1], and the infinite mixture [2], but both involve a slow inference procedure. When faster solutions are required for robust inference, and a two-component mixture is an adequate model for the task, we believe the TGP is a very attractive option.

## References

[1] Volker Tresp. Mixtures of Gaussian processes. In *Advances in Neural Information Processing Systems*, pages 654–660, 2000.

[2] Carl Edward Rasmussen and Zoubin Ghahramani. Infinite mixtures of gaussian process experts. In *Advances in Neural Information Processing Systems*, 2002.

[3] Paul Goldberg, Christopher Williams, and Christopher Bishop. Regression with input-dependent noise: a Gaussian process treatment. In *Advances in Neural Information Processing Systems*. MIT Press, 1998.

[4] Edward Snelson and Zoubin Ghahramani. Sparse Gaussian processes using pseudo-inputs. In *Advances in Neural Information Processing Systems 18*. MIT Press, 2005.

[5] Thomas Minka. *A family of algorithms for approximate Bayesian inference*. PhD thesis, Massachusetts Institute of Technology, 2001.

[6] Carl Rasmussen and Christopher Williams. *Gaussian processes for machine learning*. MIT Press, 2006.

[7] Malte Kuss. *Gaussian process models for robust regression, classification and reinforcement learning*. PhD thesis, Technische Universität Darmstadt, 2006.

[8] Matthias Seeger. Expectation propagation for exponential families, 2005. Available from `http://www.cs.berkeley.edu/~mseeger/papers/epexpfam.ps.gz`.

[9] J. H. Friedman. Multivariate adaptive regression splines. *Annals of Statistics*, 19(1):1–67, 1991.

[10] B.W. Silverman. Some aspects of the spline smoothing approach to non-parametric regression curve fitting. *Journal of the Royal Statistical Society B*, 47:1–52, 1985.

[11] Edward Snelson, Carl Edward Rasmussen, and Zoubin Ghahramani. Warped Gaussian processes. In *Advances in Neural Information Processing Systems 16*, 2003.

